# Perception of the structure of the physical world using unknown multimodal sensors and effectors

**D. Philipona**
Sony CSL, 6 rue Amyot
75005 Paris, France
david.philipona@m4x.org

**J.K. O'Regan**
Laboratoire de Psychologie Expérimentale, CNRS
Université René Descartes, 71, avenue Edouard Vaillant
92774 Boulogne-Billancourt Cedex, France
http://nivea.psycho.univ-paris5.fr

**J.-P. Nadal**
Laboratoire de Physique Statistique, ENS
rue Lhomond
75231 Paris Cedex 05

**O. J.-M. D. Coenen**
Sony CSL, 6 rue Amyot
75005 Paris, France

## Abstract

Is there a way for an algorithm linked to an unknown body to infer by itself information about this body and the world it is in? Taking the case of space for example, is there a way for this algorithm to realize that its body is in a three dimensional world? Is it possible for this algorithm to discover how to move in a straight line? And more basically: do these questions make any sense at all given that the algorithm only has access to the very high-dimensional data consisting of its sensory inputs and motor outputs?

We demonstrate in this article how these questions can be given a positive answer. We show that it is possible to make an algorithm that, by analyzing the law that links its motor outputs to its sensory inputs, discovers information about the structure of the world regardless of the devices constituting the body it is linked to. We present results from simulations demonstrating a way to issue motor orders resulting in "fundamental" movements of the body as regards the structure of the physical world.

## 1 Introduction

What is it possible to discover from behind the interface of an unknown body, embedded in an unknown world? In previous work [4] we presented an algorithm that can deduce the dimensionality of the outside space in which it is embedded, by making random movements and studying the intrinsic properties of the relation linking outgoing motor orders to

resulting changes of sensory inputs (the so called sensorimotor law [3]).

In the present article we provide a more advanced mathematical overview together with a more robust algorithm, and we also present a multimodal simulation.

The mathematical section provides a rigorous treatment, relying on concepts from differential geometry, of what are essentially two very simple ideas. The first idea is that transformations of the organism-environment system which leave the sensory inputs unchanged will do this independently of the code or the structure of sensors, and are in fact the only aspects of the sensorimotor law that are independent of the code (property 1). In a single given sensorimotor configuration the effects of such transformations induce what is called a tangent space over which linear algebra can be used to extract a small number of independent basic elements, which we call "measuring rod". The second idea is that there is a way of applying these measuring rods globally (property 2) so as to discover an overall substructure in the set of transformations that the organism-environment system can suffer, and that leave sensory inputs unchanged. Taken together these ideas make it possible, if the sensory devices are sufficiently informative, to extract an algebraic group structure corresponding to the intrinsic properties of the space in which the organism is embedded.

The simulation section is for the moment limited to an implementation of the first idea. It presents briefly the main steps of an implementation giving access to the measuring rods, and presents the results of its application to a virtual rat with mixed visual, auditory and tactile sensors (see Figure 2). The group discovered reveals the properties of the Euclidian space implicit in the equations describing the physics of the simulated world.

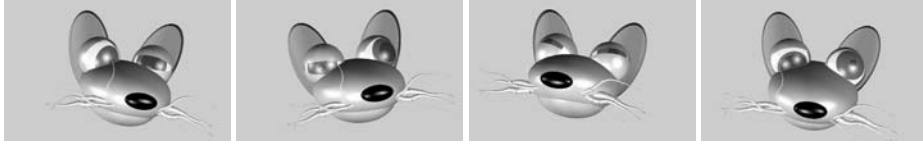

Figure 1: The virtual organism used for the simulations. Random motor commands produce random changes in the rat's body configuration, involving uncoordinated movements of the head, changes in the gaze direction, and changes in the aperture of the eyelids and diaphragms.

## 2   Mathematical formulation

Let us note $S$ the sensory inputs, and $M$ the motor outputs. They are the only things the algorithm can access. Let us note $P$ the configurations of the body controlled by the algorithm and $E$ the configurations of the environment.

We will assume that the body position is controlled by the multidimensional motor outputs through some law $\varphi_a$ and that the sensory devices together deliver a multidimensional input that is a function $\varphi_b$ of the configuration of the body and the configuration of the environment:

$$P = \varphi_a(M) \quad \text{and} \quad S = \varphi_b(P, E)$$

We shall write $\varphi(M, E) \stackrel{def}{=} \varphi_b(\varphi_a(M), E)$, note $\mathcal{S}, \mathcal{M}, \mathcal{P}, \mathcal{E}$ the sets of all $S, M, P, E$, and assume that $\mathcal{M}$ and $\mathcal{E}$ are manifolds.

## 2.1 Isotropy group of the sensorimotor law

Through time, the algorithm will be able to experiment a set of *sensorimotor laws* linking its inputs to its outputs:

$$\varphi(\cdot, \mathcal{E}) \overset{def}{=} \{M \mapsto \varphi(M, E), E \in \mathcal{E}\}$$

These are a set of functions linking $S$ to $M$, parametrized by the environmental state $E$. Our goal is to extract from this set something that does not depend on the way the sensory information is provided. In other words something that would be the same for all $h \circ \varphi(\cdot, \mathcal{E})$, where $h$ is an invertible function corresponding to a change of encoding, including changes of the sensory devices (as long as they provide access to the same information).

If we note $Sym(X) \overset{def}{=} \{f : X \to X, \ f \text{ one to one mapping}\}$, and consider :

$$\Gamma(\varphi) = \{f \in Sym(\mathcal{M} \times \mathcal{E}) \text{ such that } \varphi \circ f = \varphi\}$$

then

**Property 1** $\Gamma(\varphi_1) = \Gamma(\varphi_2) \Leftrightarrow \exists f \in Sym(\mathcal{S}) \text{ such that } \varphi_1 = f \circ \varphi_2$

Thus $\Gamma(\varphi)$ is invariant by change of encoding, and retains from $\varphi$ all that is independent of the encoding. This result is easily understood using an example from physics: think of a light sensor with unknown characteristics in a world consisting of a single point light source. The *values* of the measures are very dependent on the sensor, but the fact that they are equal on concentric spheres is an intrinsic property of the physics of the situation ($\Gamma(\varphi)$, in this case, would be the group of rotations) and is independent of the code and of the sensor's characteristics.

But how can we understand the transformations $f$ which, first, involve a manifold $\mathcal{E}$ the algorithm does not know, and second that are *invisible* since $\varphi \circ f = \varphi$. We will show that, under one reasonable assumption, there is an algorithm that can discover the Lie algebra of the Lie subgroups of $\Gamma(\varphi)$ that have independent actions over $\mathcal{M}$ and $\mathcal{E}$, i.e. Lie groups $G$ such that $g(M, E) = (g_1(M), g_2(E))$ for any $\in G$, with

$$\varphi(g_1(M), g_2(E)) = \varphi(M, E) \quad \forall g \in G \tag{1}$$

## 2.2 Fundamental vector fields over the sensory inputs

We will assume that the sensory inputs provide enough information to observe univocally the changes of the environment when the exteroceptive sensors do not move. In mathematical form, we will assume that:

**Condition 1** *There exists* $\mathcal{U} \times \mathcal{V} \subset \mathcal{M} \times \mathcal{E}$ *such that* $\varphi(M, \cdot)$ *is an injective immersion from* $\mathcal{V}$ *to* $\mathcal{S}$ *for any* $M \in \mathcal{U}$

Under this condition, $\varphi(M, \mathcal{V})$ is a manifold for any $P \in \mathcal{U}$ and $\varphi(M, \cdot)$ is a diffeomorphism from $\mathcal{V}$ to $\varphi(M, \mathcal{V})$. We shall write $\varphi^{-1}(M, \cdot)$ its inverse. Choosing $M_0 \in \mathcal{U}$, it is thus possible to define an action $\phi^{M_0}$ of $G$ over the manifold $\varphi(M_0, \mathcal{V})$ :

$$\phi^{M_0}(g, S) \overset{def}{=} \varphi(M_0, g_2(\varphi^{-1}(M_0, S))) \quad \forall S \in \varphi(M_0, \mathcal{V})$$

As a consequence (see for instance [2]), for any left invariant vector field $X$ on $G$ there is an associated fundamental vector field $X^{\mathcal{S}}$ on $\varphi(M_0, \mathcal{V})$[1] :

$$X^{\mathcal{S}}(S) \overset{def}{=} \frac{d}{dt} \phi^{M_0}(e^{-tX}, S)_{|t=0} \quad \forall S \in \varphi(M_0, \mathcal{V})$$

The key point for us is that this whole vector field can be discovered experimentally by the algorithm from one vector alone : let us suppose the algorithm knows the one vector $\frac{d}{dt}\,\phi_1(e^{-tX}, M_0)_{|t=0} \in T\mathcal{M}_{|M_0}$ (the tangent space of $\mathcal{M}$ at $M_0$), that we will call a measuring rod. Then it can construct a motor command $M_X(t)$ such that

$$M_X(0) = M_0 \quad \text{and} \quad \dot{M}_X(0) = -\frac{d}{dt}\,\phi_1(e^{-tX}, M_0)_{|t=0}$$

and observe the fundamental field, thanks to the property:

**Property 2** $X^{\mathcal{S}}(S) = \frac{d}{dt}\,\varphi(M_X(t), \varphi^{-1}(M_0, S))_{|t=0} \quad \forall\, S \in \varphi(M_0, \mathcal{V})$

Indeed the movements of the environment reveal a sub-manifold $\varphi(M_0, \mathcal{V})$ of the manifold $\mathcal{S}$ of all sensory inputs, and this means they allow to transport the sensory image of the given measuring rod over this sub-manifold : $X(S)$ is the time derivative of the sensory inputs at $t=0$ in the movement implied by the motor command $M_X$ in that configuration of the environment yielding $S$ at $t = 0$.

The fundamental vector fields are the key to our problem because [2] :

$$\left[X^{\mathcal{S}}, Y^{\mathcal{S}}\right] = [X, Y]^{\mathcal{S}}$$

where the left term uses the bracket of the vectors fields on $\varphi(M_0, \mathcal{V})$ and the right term uses the bracket in the Lie algebra of $G$. Thus clearly we can get insight into the properties of the latter by the study of these fields. If the action $\phi^{M_0}$ is effective (and it is possible to show that for any $G$ there is a subgroup such that it is),we have the additional properties:

1. $X \mapsto X^{\mathcal{S}}$ is an injective Lie algebra morphism: we can understand the whole Lie algebra of $G$ through the Lie bracket over the fundamental vector fields

2. $G$ is diffeomorphic to the group of finite compositions of fundamental flows : any element $g$ of $G$ can be written as $g = e^{X_1} e^{X_2} \ldots e^{X_k}$, and

$$\phi^{M_0}(g, S) = \phi^{M_0}(e^{X_1}, \phi^{M_0}(e^{X_2}, \ldots \phi^{M_0}(e^{X_k}, S)))$$

### 2.3 Discovery of the measuring rods

Thus the question is: how can the algorithm come to know the measuring rods? If $\varphi$ is not singular (that is: is a subimmersion on $\mathcal{U} \times \mathcal{V}$, see [1]), then it can be demonstrated that:

**Property 3** $\frac{\partial \varphi}{\partial M}(M_0, E_0)\left[\dot{M} - \dot{M}_X\right] = 0 \Rightarrow \frac{d}{dt}\,\varphi(M(t), \cdot)_{|t=0} = X^{\mathcal{S}}(\varphi(M_0, \cdot))$

This means that the particular choice of one vector of $T\mathcal{M}_{|M_0}$ among those that have the same sensory image as a given measuring rod is of no importance for the construction of the associated vector field. Consequently, the search for the measuring rods becomes the search for their sensory image, which form a linear subspace of the intersection of the tangent spaces of $\varphi(M_0, \mathcal{V})$ and $\varphi(\mathcal{U}, E_0)$ (as a direct consequence of property 2):

$$\forall X \quad \frac{\partial \varphi}{\partial M}(M_0, E_0)\frac{d}{dt}\,\phi_1(e^{-tX}, M_0)_{|t=0} \in T\varphi(M_0, \mathcal{V})_{|S_0} \bigcap T\varphi(\mathcal{U}, E_0)_{|S_0}$$

But what about the rest of the intersection? Reciprocally, it can be shown that:

**Property 4** *Any measuring rod that has a sensory image in the intersection of the tangent spaces of $\varphi(M_0, \mathcal{V})$ and $\varphi(\mathcal{U}, E)$ for any $E \in \mathcal{V}$ reveals a monodimensional subgroup of transformations over $\mathcal{V}$ that is invariant under any change of encoding.*

# 3   Simulation

## 3.1   Description of the virtual rat

We have applied these ideas to a virtual body satisfying the different necessary conditions for the theory to be applied. Though our approach would also apply to the situation where the sensorimotor law involves time-varying functions, for simplicity here we shall take the restricted case where $S$ and $M$ are linked by a non-delayed relationship. We thus implemented a rat's head with instantaneous reactions so that $M \in \mathcal{R}^m$ and $S \in \mathcal{R}^s$. In the simulation, $m$ and $s$ have been arbitrarily assigned the value $300$.

The head had visual, auditory and tactile input devices (see Figure 2). The visual device consisted of two eyes, each one being constituted by $40$ photosensitive cells randomly distributed on a planar retina, one lens, one diaphragm (or pupil) and two eyelids. The images of the 9 light sources constituting the environment were projected through the lens on the retina to locally stimulate photosensitive cells, with a total influx related to the aperture of the diaphragm and the eyelids. The auditory device was constituted by one amplitude sensor in each of the two ears, with a sensitivity profile favoring auditory sources with azimuth and elevation $0°$ with respect to the orientation of the head. The tactile device was constituted by $4$ whiskers on each side of the rat's jaw, that stuck to an object when touching it, and delivered a signal related to the shift from rest position. The global sensory inputs of dimension 90 ($2 \times 40$ photosensors plus 2 auditory sensors plus 8 tactile sensors) were delivered to the algorithm through a linear mixing of all the signals delivered by these sensors, using a random matrix $W_S \in \mathcal{M}(s, 90)$ representing some sensory neural encoding in dimension $s = 300$.

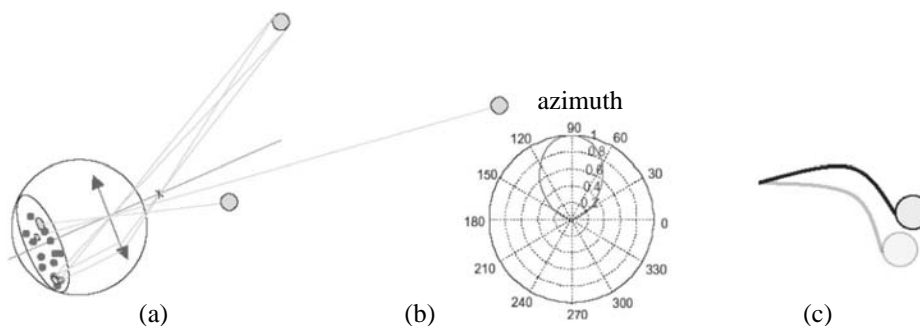

<div style="text-align:center">(a)&emsp;&emsp;&emsp;&emsp;&emsp;&emsp;(b)&emsp;&emsp;&emsp;&emsp;&emsp;&emsp;(c)</div>

Figure 2: The sensory system. (a) the sensory part of both eyes is constituted of randomly distributed photosensitive cells (small dark dots). (b) the auditory sensors have a gain profile favoring sounds coming from the front of the ears. (c) tactile devices stick to the sources they come into contact with.

The motor device was as follows. Sixteen control parameters were constructed from linear combinations of the motor outputs of dimension $m = 300$ using a random matrix $W_M \in \mathcal{M}(16, m)$ representing some motor neural code. The configuration of the rat's head was then computed from these sixteen variables in this way: six parameters controlled the position and orientation of the head, and, for each eye, three controlled the eye orientation plus two the aperture of the diaphragm and the eyelids. The whiskers were not controllable, but were fixed to the head.

In the simulation we used linear encoding $W_S$ and $W_M$ in order to show that the algorithm worked even when the dimension of the sensory and motor vectors was high. Note first however that any, even non-linear, continuous high-dimensional function could have been used instead of the linear mixing matrices. More important, note that even when linear

mixing is used, the sensorimotor law is highly *nonlinear*: the sensors deliver signals that are not linear with respect to the configuration of the rat's head, and this configuration is itself not linear with respect to the motor outputs.

## 3.2 The algorithm

The first important result of the mathematical section was that the sensory images of the measuring rods are in the intersection between the tangent space of the sensory inputs observed when issuing different motor outputs while the environment is immobile, and the tangent space of the sensory inputs observed when the command being issued is constant.

In the present simulation we will only be making use of this point, but keep in mind that the second important result was the relation between the fundamental vector fields and these measuring rods. This implies that the tangent vectors we are going to find by an experiment for a given sensory input $S_0 = \varphi(M_0, E_0)$ can be transported in a particular way over the whole sub-manifold $\varphi(M_0, \mathcal{V})$, thereby generating the sensory consequences of any transformation of $\mathcal{E}$ associated with the Lie subgroup of $\Gamma(\varphi)$ whose measuring rods have been found.

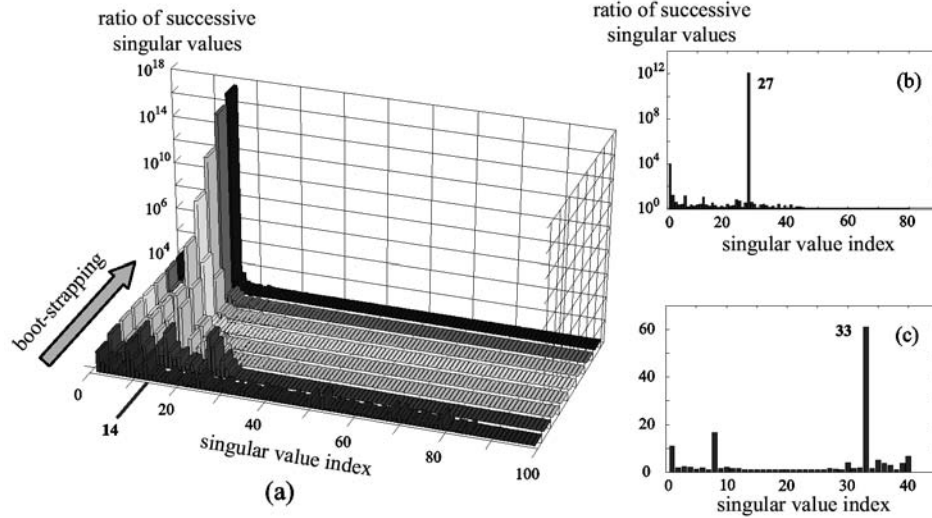

Figure 3: Amplitudes of the ratio of successive singular values of : (a) the estimated tangent sensorimotor law (when $E$ is fixed at $E_0$) during the bootstrapping process; (b) the matrix corresponding to an estimated generating family for the tangent space to the manifold of sensory inputs observed when $M$ is fixed at $M_0$; (c) the matrix constituted by concatenating the vectors found in the two previous cases. The nullspaces of the two first matrices reflect redundant variables; the nullspace of the last one is related to the intersection of the two first tangent spaces (see equation 2). The graphs show there are 14 control parameters with respect to the body, and 27 variables to parametrize the environment (see text). The nullspace of the last matrix leads to the computation of an intersection of dimension 6 reflecting the Lie group of Euclidian transformations $SE(3)$ (see text).

In [4], the simulation aimed to demonstrate that the dimensions of the different vector spaces involved were accessible. We now present a simulation that goes beyond this by estimating these vector space themselves, in particular $T\varphi(M_0, \mathcal{V})_{|S_0} \bigcap T\varphi(\mathcal{U}, E_0)_{|S_0}$, in the case of multimodal sensory inputs and with a robust algorithm. The method previously used to estimate the first tangent space, and more specifically its dimension, indeed required

an unrealistic level of accuracy. One of the reasons was the poor behavior of the Singular Value Decomposition when dealing with badly conditioned matrices. We have developed a much more stable method, that furthermore uses time derivatives as a more plausible way to estimate the differential than multivariate linear approximation. Indeed, the nonlinear functional relationship between the motor output and the sensory inputs implies an exact linear relationship between their respective time derivative at a given motor output $M_0$

$$S(t) = \varphi(M(t), E_0) \Rightarrow \dot{S}(0) = \frac{\partial \varphi}{\partial M}(M_0, E_0)\dot{M}(0)$$

and this linear relationship can be estimated as the linear mapping associating the $\dot{M}(0)$, for any curve in the motor command space such that $M(0) = M_0$, to the resulting $\dot{S}(0)$. The idea is then to use bootstrapping to estimate the time derivative of the "good" sensory input combinations along the "good" movements so that this linear relation is diagonal and the decomposition unnecessary : the purpose of the SVD used at each step is to provide an indication of what vectors seem to be of interest. At the end of the process, when the linear relationship is judged to be sufficiently diagonal, the singular values are taken as the diagonal elements, and are thus estimated with the precision of the time derivative estimator. Figure 3a presents the evolution of the estimated dimension of the tangent space during this bootstrapping process.

Using this method in the first stage of the experiment when the environment is immobile makes it possible for the algorithm, at the same time as it finds a basis for the tangent space, to calibrate the signals coming from the head : it extracts sensory input combinations that are meaningful as regards its own mobility. Then during a second stage, using these combinations, it estimates the tangent space to sensory inputs resulting from movement of the environment while it keeps its motor output fixed at $M_0$. Finally, using the tangent spaces estimated in these two stages, it computes their intersection : if $TS_M$ is a matrix containing the basis of the first tangent space, and $TS_E$ a basis of the second tangent space, then the nullspace of $[TS_M, TS_E]$ allows to generate the intersection of the two spaces:

$$[TS_M, TS_E]\lambda = 0 \Rightarrow TS_M\lambda_M = -TS_E\lambda_E \quad \text{where } \lambda = (\lambda_M^T, \lambda_E^T)^T \qquad (2)$$

To conclude, using the pseudo-inverse of the tangent sensorimotor law, the algorithm computes measuring rods that have a sensory image in that intersection; and this computation is simple since the adaptation process made the tangent law diagonal.

### 3.3 Results[2]

Figure 3a demonstrates the evolution of the estimation of the ratio between successive singular values. The maximum of this ratio can be taken as the frontier between significantly non-zero values and zero ones, and thus reveals the dimension of the tangent space to the sensory inputs observed in an immobile environment. There are indeed 14 effective parameters of control of the body with respect to the sensory inputs: from the 16 parameters described in section 3.1, for each eye the two parameters controlling the aperture of the diaphragm and the eyelids combine in a single effective one characterizing the total incoming light influx.

After this adaptation process the tangent space to sensory inputs observed for a fixed motor output $M_0$ can be estimated without bootstrapping as shown, as regards its dimension ($27 = 9 \times 3$ for the 9 light sources moving in a three dimensional space), in Figure 3b. The intersection is computed from the nullspace of the matrix constituted by concatenation of generating vectors of the two previous spaces, using equation 2. This nullspace is of

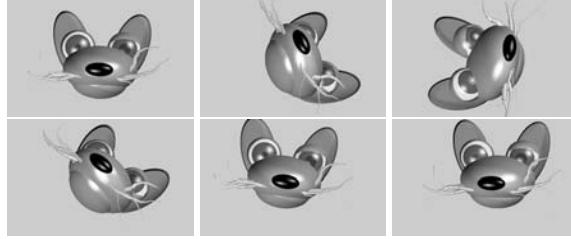

Figure 4: The effects of motor commands corresponding to a generating family of 6 independent measuring rods computed by the algorithm. They reveal the control of the head in a rigid fashion. Without the Lie bracket to understand commutativity, these movements involve arbitrary compositions of translations and rotations.

dimension $41 - 35 = 6$, as shown in Figure 3c. Note that the graph shows the *ratio* of successive singular values, and thus has one less value than the number of vectors. Figure 4 demonstrates the movements of the rat's head associated with the measuring rods found using the pseudoinverse of the sensorimotor law. Contrast these with the non-rigid movements of the rat's head associated with random motor commands of Figure 1.

## 4    Conclusion

We have shown that sensorimotor laws possess intrinsic properties related to the structure of the physical world in which an organism's body is embedded. These properties have an overall group structure, for which smoothly parametrizable subgroups that act separately on the body and on the environment can be discovered. We have briefly presented a simulation demonstrating the way to access the measuring rods of these subgroups.

We are currently conducting our first successful experiments on the estimation of the Lie bracket. This will allow the groups whose measuring rods have been found to be decomposed. It will then be possible for the algorithm to distinguish for instance between translations and rotations, and between rotations around different centers.

The question now is to determine what can be done with these first results: is this intrinsic understanding of space enough to discover the subgroups of $\Gamma(\varphi)$ that do not act both on the body and the environment: for example those acting on the body alone should provide a decomposition of the body with respect to its articulations.

The ultimate goal is to show that there is a way of extracting *objects* in the environment from the sensorimotor law, even though nothing is known about the sensors and effectors.

## Footnotes

[1] To avoid heavy notations we have written $X^{\mathcal{S}}$ instead of $X^{\varphi(M_0, \mathcal{V})}$.

[2]The Matlab code of the simulation can be downloaded at `http://nivea.psycho.univ-paris5.fr/~philipona` for further examination.

## References

[1] N. Bourbaki. *Variétes différentielles et analytiques. Fascicule de résultats.* Hermann, 1971-1997.

[2] T. Masson. *Géométrie différentielle, groupes et algèbres de Lie, fibrés et connexions.* LPT, 2001.

[3] J. K. O'Regan and A. Noë. A sensorimotor account of vision and visual consciousness. *Behavioral and Brain Sciences*, 24(5), 2001.

[4] D. Philipona, K. O'Regan, and J.-P. Nadal. Is there something out there ? Inferring space from sensorimotor dependencies. *Neural Computation*, 15(9), 2003.
